# Complex Inference in Neural Circuits with Probabilistic Population Codes and Topic Models

**Jeff Beck**
Department of Brain and Cognitive Sciences
University of Rochester
jbeck@bcs.rochester.edu

**Katherine Heller**
Department of Statistical Science
Duke University
kheller@stat.duke.edu

**Alexandre Pouget**
Department of Neuroscience
University of Geneva
Alexandre.Pouget@unige.ch

## Abstract

Recent experiments have demonstrated that humans and animals typically reason probabilistically about their environment. This ability requires a neural code that represents probability distributions and neural circuits that are capable of implementing the operations of probabilistic inference. The proposed probabilistic population coding (PPC) framework provides a statistically efficient neural representation of probability distributions that is both broadly consistent with physiological measurements and capable of implementing some of the basic operations of probabilistic inference in a biologically plausible way. However, these experiments and the corresponding neural models have largely focused on simple (tractable) probabilistic computations such as cue combination, coordinate transformations, and decision making. As a result it remains unclear how to generalize this framework to more complex probabilistic computations. Here we address this short coming by showing that a very general approximate inference algorithm known as Variational Bayesian Expectation Maximization can be naturally implemented within the linear PPC framework. We apply this approach to a generic problem faced by any given layer of cortex, namely the identification of latent causes of complex mixtures of spikes. We identify a formal equivalent between this spike pattern demixing problem and topic models used for document classification, in particular Latent Dirichlet Allocation (LDA). We then construct a neural network implementation of variational inference and learning for LDA that utilizes a linear PPC. This network relies critically on two non-linear operations: divisive normalization and super-linear facilitation, both of which are ubiquitously observed in neural circuits. We also demonstrate how online learning can be achieved using a variation of Hebb's rule and describe an extension of this work which allows us to deal with time varying and correlated latent causes.

## 1   Introduction to Probabilistic Inference in Cortex

Probabilistic (Bayesian) reasoning provides a coherent and, in many ways, optimal framework for dealing with complex problems in an uncertain world. It is, therefore, somewhat reassuring that behavioural experiments reliably demonstrate that humans and animals behave in a manner consistent with optimal probabilistic reasoning when performing a wide variety of perceptual [1, 2, 3], motor [4, 5, 6], and cognitive tasks[7]. This remarkable ability requires a neural code that represents probability distribution functions of task relevant stimuli rather than just single values. While there

are many ways to represent functions, Bayes rule tells us that when it comes to probability distribution functions, there is only one statistically optimal way to do it. More precisely, Bayes Rule states that any pattern of activity, $\mathbf{r}$, that *efficiently* represents a probability distribution over some task relevant quantity $s$, must satisfy the relationship $p(s|\mathbf{r}) \propto p(\mathbf{r}|s)p(s)$, where $p(\mathbf{r}|s)$ is the stimulus conditioned likelihood function that specifies the form of neural variability, $p(s)$ gives the prior belief regarding the stimulus, and $p(s|\mathbf{r})$ gives the posterior distribution over values of the stimulus, $s$ given the representation $\mathbf{r}$ . Of course, it is unlikely that the nervous system consistently achieves this level of optimality. None-the-less, Bayes rule suggests the existence of a link between neural variability as characterized by the likelihood function $p(\mathbf{r}|s)$ and the state of belief of a mature statistical learning machine such as the brain.

The so called Probabilistic Population Coding (or PPC) framework[8, 9, 10] takes this link seriously by proposing that the function encoded by a pattern of neural activity $\mathbf{r}$ is, in fact, the likelihood function $p(\mathbf{r}|s)$. When this is the case, the precise form of the neural variability informs the nature of the neural code. For example, the exponential family of statistical models with linear sufficient statistics has been shown to be flexible enough to model the first and second order statistics of *in vivo* recordings in awake behaving monkeys[9, 11, 12] and anesthetized cats[13]. When the likelihood function is modeled in this way, the log posterior probability over the stimulus is linearly encoded by neural activity, i.e.

$$\log p(s|\mathbf{r}) = \mathbf{h}(s) \cdot \mathbf{r} - \log Z(\mathbf{r}) \tag{1}$$

Here, the stimulus dependent kernel, $\mathbf{h}(s)$, is a vector of functions of $s$, the dot represents a standard dot product, and $Z(\mathbf{r})$ is the partition function which serves to normalize the posterior. This log linear form for a posterior distribution is highly computationally convenient and allows for evidence integration to be implemented via linear operations on neural activity[14, 8].

Proponents of this kind of **linear** PPC have demonstrated how to build biologically plausible neural networks capable of implementing the operations of probabilistic inference that are needed to optimally perform the behavioural tasks listed above. This includes, linear PPC implementations of cue combination[8], evidence integration over time, maximum likelihood and maximum a posterior estimation[9], coordinate transformation/auditory localization[10], object tracking/Kalman filtering[10], explaining away[10], and visual search[15]. Moreover, each of these neural computations has required only a single recurrently connected layer of neurons that is capable of just two non-linear operations: coincidence detection and divisive normalization, both of which are widely observed in cortex[16, 17].

Unfortunately, this research program has been a piecemeal effort that has largely proceeded by building neural networks designed deal with particular problems. As a result, there have been no proposals for a general principle by which neural network implementations of linear PPCs might be generated and no suggestions regarding how to deal with complex (intractable) problems of probabilistic inference.

In this work, we will partially address this short coming by showing that Variation Bayesian Expectation Maximization (VBEM) algorithm provides a general scheme for approximate inference and learning with linear PPCs. In section 2, we briefly review the VBEM algorithm and show how it naturally leads to a linear PPC representation of the posterior as well as constraints on the neural network dynamics which build that PPC representation. Because this section describes the VB-PPC approach rather abstractly, the remainder of the paper is dedicated to concrete applications. As a motivating example, we consider the problem of inferring the concentrations of odors in an olfactory scene from a complex pattern of spikes in a population of olfactory receptor neurons (ORNs). In section 3, we argue that this requires solving a spike pattern demixing problem which is indicative of the generic problem faced by many layers of cortex. We then show that this demixing problem is equivalent to the problem addressed by a class of models for text documents know as *probabilistic topic models*, in particular Latent Dirichlet Allocation or LDA[18].

In section 4, we apply the VB-PPC approach to build a neural network implementation of probabilistic inference and learning for LDA. This derivation shows that causal inference with linear PPC's also critically relies on divisive normalization. This result suggests that this particular non-linearity may be involved in very general and fundamental probabilistic computation, rather than simply playing a role in gain modulation. In this section, we also show how this formulation allows for a probabilistic treatment of learning and show that a simple variation of Hebb's rule can implement Bayesian learning in neural circuits.

We conclude this work by generalizing this approach to time varying inputs by introducing the Dynamic Document Model (DDM) which can infer short term fluctuations in the concentrations of individual topics/odors and can be used to model foraging and other tracking tasks.

## 2  Variational Bayesian Inference with linear Probabilistic Population Codes

Variational Bayesian (VB) inference refers to a class of deterministic methods for approximating the intractable integrals which arise in the context of probabilistic reasoning. Properly implemented it can result a fast alternative to sampling based methods of inference such as MCMC[19] sampling. Generically, the goal of any Bayesian inference algorithm is to infer a posterior distribution over behaviourally relevant latent variables $\mathbf{Z}$ given observations $\mathbf{X}$ and a generative model which specifies the joint distribution $p(\mathbf{X}, \mathbf{\Theta}, \mathbf{Z})$. This task is confounded by the fact that the generative model includes latent parameters $\mathbf{\Theta}$ which must be marginalized out, i.e. we wish to compute,

$$p(\mathbf{Z}|\mathbf{X}) \propto \int p(\mathbf{X}, \mathbf{\Theta}, \mathbf{Z}) d\mathbf{\Theta} \tag{2}$$

When the number of latent parameters is large this integral can be quite unwieldy. The VB algorithms simplify this marginalization by approximating the complex joint distribution over behaviourally relevant latents and parameters, $p(\mathbf{\Theta}, \mathbf{Z}|\mathbf{X})$, with a distribution $q(\mathbf{\Theta}, \mathbf{Z})$ for which integrals of this form are easier to deal with in some sense. There is some art to choosing the particular form for the approximating distribution to make the above integral tractable, however, a factorized approximation is common, i.e. $q(\mathbf{\Theta}, \mathbf{Z}) = q_{\mathbf{\Theta}}(\mathbf{\Theta}) q_{\mathbf{Z}}(\mathbf{Z})$.

Regardless, for any given observation $\mathbf{X}$, the approximate posterior is found by minimizing the Kullback-Leibler divergence between $q(\mathbf{\Theta}, \mathbf{Z})$ and $p(\mathbf{\Theta}, \mathbf{Z}|\mathbf{X})$. When a factorized posterior is assumed, the Variational Bayesian Expectation Maximization (VBEM) algorithm finds a local minimum of the KL divergence by iteratively updating, $q_{\mathbf{\Theta}}(\mathbf{\Theta})$ and $q_{\mathbf{Z}}(\mathbf{Z})$ according to the scheme

$$\log q_{\mathbf{\Theta}}^n(\mathbf{\Theta}) \sim \langle \log p(\mathbf{X}, \mathbf{\Theta}, \mathbf{Z}) \rangle_{q_{\mathbf{Z}}^n(\mathbf{Z})} \quad \text{and} \quad \log q_{\mathbf{Z}}^{n+1}(\mathbf{Z}) \sim \langle \log p(\mathbf{X}, \mathbf{\Theta}, \mathbf{Z}) \rangle_{q_{\mathbf{\Theta}}^n(\mathbf{\Theta})} \tag{3}$$

Here the brackets indicate an expected value taken with respect to the subscripted probability distribution function and the tilde indicates equality up to a constant which is independent of $\mathbf{\Theta}$ and $\mathbf{Z}$. The key property to note here is that the approximate posterior which results from this procedure is in an exponential family form and is therefore representable by a linear PPC (Eq. 1). This feature allows for the straightforward construction of networks which implement the VBEM algorithm with linear PPC's in the following way. If $\mathbf{r}_{\mathbf{\Theta}}^n$ and $\mathbf{r}_{\mathbf{Z}}^n$ are patterns of activity that use a linear PPC representation of the relevant posteriors, then

$$\log q_{\mathbf{\Theta}}^n(\mathbf{\Theta}) \sim \mathbf{h}_{\mathbf{\Theta}}(\mathbf{\Theta}) \cdot \mathbf{r}_{\mathbf{\Theta}}^n \quad \text{and} \quad \log q_{\mathbf{Z}}^{n+1}(\mathbf{Z}) \sim \mathbf{h}_{\mathbf{Z}}(\mathbf{Z}) \cdot \mathbf{r}_{\mathbf{Z}}^{n+1}. \tag{4}$$

Here the stimulus dependent kernels $\mathbf{h}_{\mathbf{Z}}(\mathbf{Z})$ and $\mathbf{h}_{\mathbf{\Theta}}(\mathbf{\Theta})$ are chosen so that their outer product results in a basis that spans the function space on $\mathbf{Z} \times \mathbf{\Theta}$ given by $\log p(\mathbf{X}, \mathbf{\Theta}, \mathbf{Z})$ for every $\mathbf{X}$. This choice guarantees that there exist functions $\mathbf{f}_{\mathbf{\Theta}}(\mathbf{X}, \mathbf{r}_{\mathbf{Z}}^n)$ and $\mathbf{f}_{\mathbf{Z}}(\mathbf{X}, \mathbf{r}_{\mathbf{\Theta}}^n)$ such that

$$\mathbf{r}_{\mathbf{\Theta}}^n = \mathbf{f}_{\mathbf{\Theta}}(\mathbf{X}, \mathbf{r}_{\mathbf{Z}}^n) \quad \text{and} \quad \mathbf{r}_{\mathbf{Z}}^{n+1} = \mathbf{f}_{\mathbf{Z}}(\mathbf{X}, \mathbf{r}_{\mathbf{\Theta}}^n) \tag{5}$$

satisfy Eq. 3. When this is the case, simply iterating the discrete dynamical system described by Eq. 5 until convergence will find the VBEM approximation to the posterior. This is one way to build a neural network implementation of the VB algorithm. However, its not the only way. In general, any dynamical system which has stable fixed points in common with Eq. 5 can also be said to implement the VBEM algorithm. In the example below we will take advantage of this flexibility in order to build biologically plausible neural network implementations.

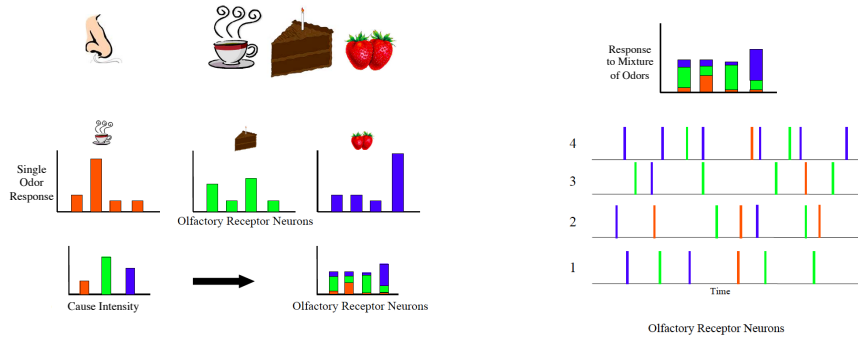

Figure 1: (Left) Each cause (e.g. coffee) in isolation results in a pattern of neural activity (top). When multiple causes contribute to a scene this results in an overall pattern of neural activity which is a mixture of these patterns weighted by the intensities (bottom). (Right) The resulting pattern can be represented by a raster, where each spike is colored by its corresponding latent cause.

# 3 Probabilistic Topic Models for Spike Train Demixing

Consider the problem of odor identification depicted in Fig. 1. A typical mammalian olfactory system consists of a few hundred different types of olfactory receptor neurons (ORNs), each of which responds to a wide range of volatile chemicals. This results in a highly distributed code for each odor. Since, a typical olfactory scene consists of many different odors at different concentrations, the pattern of ORN spike trains represents a complex mixture. Described in this way, it is easy to see that the problem faced by early olfactory cortex can be described as the task of demixing spike trains to infer latent causes (odor intensities).

In many ways this olfactory problem is a generic problem faced by each cortical layer as it tries to make sense of the activity of the neurons in the layer below. The input patterns of activity consist of spikes (or spike counts) labeled by the axons which deliver them and summarized by a histogram which indicates how many spikes come from each input neuron. Of course, just because a spike came from a particular neuron does not mean that it had a particular cause, just as any particular ORN spike could have been caused by any one of a large number of volatile chemicals. Like olfactory codes, cortical codes are often distributed and multiple latent causes can be present at the same time.

Regardless, this spike or histogram demixing problem is formally equivalent to a class of demixing problems which arise in the context of probabilistic topic models used for document modeling. A simple but successful example of this kind of topic model is called Latent Dirichlet Allocation (LDA) [18]. LDA assumes that word order in documents is irrelevant and, therefore, models documents as histograms of word counts. It also assumes that there are $K$ topics and that each of these topics appears in different proportions in each document, e.g. 80% of the words in a document might be concerned with coffee and 20% with strawberries. Words from a given topic are themselves drawn from a distribution over words associated with that topic, e.g. when talking about coffee you have a 5% chance of using the word 'bitter'. The goal of LDA is to infer both the distribution over topics discussed in each document and the distribution of words associated with each topic. We can map the generative model for LDA onto the task of spike demixing in cortex by letting topics become latent causes or odors, words become neurons, word occurrences become spikes, word distributions associated with each topic become patterns of neural activity associated with each cause, and different documents become the observed patterns of neural activity on different trials. This equivalence is made explicit in Fig. 2 which describes the standard generative model for LDA applied to documents on the left and mixtures of spikes on the right.

# 4 LDA Inference and Network Implementation

In this section we will apply the VB-PPC formulation to build a biologically plausible network capable of approximating probabilistic inference for spike pattern demixing. For simplicity, we will use the equivalent Gamma-Poisson formulation of LDA which directly models word and topic counts

1. For each topic $k = 1, \ldots, K$,
    (a) Distribution over words
        $\beta_k \sim \text{Dirichlet}(\eta_0)$
2. For document $d = 1, \ldots, D$,
    (a) Distribution over topics
        $\theta_d \sim \text{Dirichlet}(\alpha_0)$
    (b) For word $m = 1, \ldots, \Omega_d$
        i. Topic assignment
            $z_{d,m} \sim \text{Multinomial}(\theta_d)$
        ii. Word assignment
            $\omega_{d,m} \sim \text{Multinomial}(\beta_{z_m})$

1. For latent cause $k = 1, \ldots, K$,
    (a) Pattern of neural activity
        $\beta_k \sim \text{Dirichlet}(\eta_0)$
2. For scene $d = 1, \ldots, D$,
    (a) Relative intensity of each cause
        $\theta_d \sim \text{Dirichlet}(\alpha_0)$
    (b) For spike $m = 1, \ldots, \Omega_d$
        i. Cause assignment
            $z_{d,m} \sim \text{Multinomial}(\theta_d)$
        ii. Neuron assignment
            $\omega_{d,m} \sim \text{Multinomial}(\beta_{z_m})$

Figure 2: (Left) The LDA generative model in the context of document modeling. (Right) The corresponding LDA generative model mapped onto the problem of spike demixing. Text related attributes on the left, in red, have been replaced with neural attributes on the right, in green.

rather than topic assignments. Specifically, we define, $R_{d,j}$ to be the number of times neuron $j$ fires during trial $d$. Similarly, we let $N_{d,j,k}$ to be the number of times a spike in neuron $j$ comes from cause $k$ in trial $d$. These new variables play the roles of the cause and neuron assignment variables, $z_{d,m}$ and $\omega_{d,m}$ by simply counting them up. If we let $c_{d,k}$ be an un-normalized intensity of cause $j$ such that $\theta_{d,k} = c_{d,k} / \sum_k c_{d,k}$ then the generative model,

$$R_{d,j} = \sum_k N_{d,j,k} \qquad N_{d,j,k} \sim \text{Poisson}(\beta_{j,k} c_{d,k}) \qquad c_{d,k} \sim \text{Gamma}(\alpha_k^0, C^{-1}). \qquad (6)$$

is equivalent to the topic models described above. Here the parameter $C$ is a scale parameter which sets the expected total number of spikes from the population on each trial. Note that, the problem of inferring the $w_{j,k}$ and $c_{d,k}$ is a non-negative matrix factorization problem similar to that considered by Lee and Seung[20]. The primary difference is that, here, we are attempting to infer a probability distribution over these quantities rather than maximum likelihood estimates. See supplement for details. Following the prescription laid out in section 2, we approximate the posterior over latent variables given a set of input patterns, $\mathbf{R}_d, d = 1, \ldots, D$, with a factorized distribution of the form, $q_{\mathbf{N}}(\mathbf{N}) q_{\mathbf{c}}(\mathbf{c}) q_{\boldsymbol{\beta}}(\boldsymbol{\beta})$. This results in marginal posterior distributions $q\left(\beta_{:,k} | \eta_{:,k}\right)$, $q\left(c_{d,k} | \alpha_{d,k}, C^{-1} + 1)\right)$, and $q\left(N_{d,j,:} | \log p_{d,j,:}, R_{d,i}\right)$ which are Dirichlet, Gamma, and Multinomial respectively. Here, the parameters $\eta_{:,k}$, $\alpha_{d,k}$, and $\log p_{d,j,:}$ are the natural parameters of these distributions. The VBEM update algorithm yields update rules for these parameters which are summarized in Fig. 3 Algorithm1.

---

**Algorithm 1:** Batch VB updates

1: **while** $\eta_{j,k}$ not converged **do**
2:    **for** $d = 1, \cdots, D$ **do**
3:       **while** $p_{d,j,k}, \alpha_{d,k}$ not converged **do**
4:          $\alpha_{d,k} \rightarrow \alpha_0 + \sum_j R_{d,j} p_{d,j,k}$
5:          $p_{d,j,k} \rightarrow$
         $\frac{\exp\left(\psi(\eta_{j,k}) - \psi(\bar{\eta}_k)\right) \exp \psi(\alpha_{d,k})}{\sum_i \exp\left(\psi(\eta_{j,i}) - \psi(\bar{\eta}_i)\right) \exp \psi(\alpha_{d,i})}$
6:       **end while**
7:    **end for**
8:    $\eta_{j,k} = \eta^0 + \sum_d R_{d,j} p_{d,j,k}$
9: **end while**

**Algorithm 2:** Online VB updates

1: **for** $d = 1, \cdots, D$ **do**
2:    reinitialize $p_{j,k}, \alpha_k \,\forall j, k$
3:    **while** $p_{j,k}, \alpha_k$ not converged **do**
4:       $\alpha_k \rightarrow \alpha_0 + \sum_j R_{d,j} p_{j,k}$
5:       $p_{j,k} \rightarrow$
      $\frac{\exp\left(\psi(\eta_{j,k}) - \psi(\bar{\eta}_k)\right) \exp \psi(\alpha_k)}{\sum_i \exp\left(\psi(\eta_{j,i}) - \psi(\bar{\eta}_i)\right) \exp \psi(\alpha_i)}$
6:    **end while**
7:    $\eta_{j,k} \rightarrow$
   $(1 - dt)\eta_{j,k} + dt(\eta^0 + R_{d,j} p_{j,k})$
8: **end for**

---

Figure 3: Here $\bar{\eta}_k = \sum_j \eta_{j,k}$ and $\psi(x)$ is the digamma function so that $\exp \psi(x)$ is a smoothed threshold linear function.

Before we move on to the neural network implementation, note that this standard formulation of variational inference for LDA utilizes a batch learning scheme that is not biologically plausible. Fortunately, an online version of this variational algorithm was recently proposed and shown to give

superior results when compared to the batch learning algorithm[21]. This algorithm replaces the sum over $d$ in update equation for $\eta_{j,k}$ with an incremental update based upon only the most recently observed pattern of spikes. See Fig. 3 Algorithm 2.

## 4.1 Neural Network Implementation

Recall that the goal was to build a neural network that implements the VBEM algorithm for the underlying latent causes of a mixture of spikes using a neural code that represents the posterior distribution via a linear PPC. A linear PPC represents the natural parameters of a posterior distribution via a linear operation on neural activity. Since the primary quantity of interest here is the posterior distribution over odor concentrations, $q_{\mathbf{c}}(\mathbf{c}|\boldsymbol{\alpha})$, this means that we need a pattern of activity $\mathbf{r}_{\boldsymbol{\alpha}}$ which is linearly related to the $\alpha_k$'s in the equations above. One way to accomplish this is to simply assume that the firing rates of output neurons are equal to the positive valued $\alpha_k$ parameters.

Fig. 4 depicts the overall network architecture. Input patterns of activity, $\mathbf{R}$, are transmitted to the synapses of a population of output neurons which represent the $\alpha_k$'s. The output activity is pooled to form an un-normalized prediction of the activity of each input neuron, $\bar{R}_j$, given the output layer's current state of belief about the latent causes of the $R_j$. The activity at each synapse targeted by input neuron $j$ is then inhibited divisively by this prediction. This results in a dendrite that reports to the soma a quantity, $\bar{N}_{j,k}$, which represents the fraction of unexplained spikes from input neuron $j$ that could be explained by latent cause $k$. A continuous time dynamical system with this feature and the property that it shares its fixed points with the LDA algorithm is given by

$$\frac{d}{dt}\bar{N}_{j,k} = w_{j,k}R_j - \bar{R}_j\bar{N}_{j,k} \tag{7}$$

$$\frac{d}{dt}\alpha_k = \exp\left(\psi\left(\bar{\eta}_k\right)\right)\left(\alpha_0 - \alpha_k\right) + \exp\left(\psi\left(\alpha_k\right)\right)\sum_i \bar{N}_{j,k} \tag{8}$$

where $\bar{R}_j = \sum_k w_{j,k}\exp\left(\psi\left(\alpha_k\right)\right)$, and $w_{j,k} = \exp\left(\psi\left(\eta_{j,k}\right)\right)$. Note that, despite its form, it is Eq. 7 which implements the required divisive normalization operation since, in the steady state, $\bar{N}_{j,k} = w_{j,k}R_j/\bar{R}_j$.

Regardless, this network has a variety of interesting properties that align well with biology. It predicts that a balance of excitation and inhibition is maintained in the dendrites via divisive normalization and that the role of inhibitory neurons is to predict the input spikes which target individual dendrites. It also predicts superlinear facilitation. Specifically, the final term on the right of Eq. 8 indicates that more active cells will be more sensitive to their dendritic inputs. Alternatively, this could be implemented via recurrent excitation at the population level. In either case, this is the mechanism by which the network implements a sparse prior on topic concentrations and stands in stark contrast to the winner take all mechanisms which rely on competitive mutual inhibition mechanisms. Additionally, the $\bar{\eta}_j$ in Eq. 8 represents a cell wide 'leak' parameter that indicates that the total leak should be roughly proportional to the sum total weight of the synapses which drive the neuron. This predicts that cells that are highly sensitive to input should also decay back to baseline more quickly. This implementation also predicts Hebbian learning of synaptic weights. To observe this fact, note that the online update rule for the $\eta_{j,k}$ parameters can be implemented by simply correlating the activity at each synapse, $\bar{N}_{j,k}$ with activity at the soma $\alpha_j$ via the equation:

$$\tau_L\frac{d}{dt}w_{j,k} = \exp\left(\psi\left(\bar{\eta}_k\right)\right)\left(\eta_0 - 1/2 - w_{j,k}\right) + \bar{N}_{j,k}\exp\psi\left(\alpha_k\right) \tag{9}$$

where $\tau_L$ is a long time constant for learning and we have used the fact that $\exp\left(\psi\left(\eta_{jk}\right)\right) \approx \eta_{jk} - 1/2$ for $x > 1$. For a detailed derivation see the supplementary material.

## 5 Dynamic Document Model

LDA is a rather simple generative model that makes several unrealistic assumptions about mixtures of sensory and cortical spikes. In particular, it assumes both that there are no correlations between the

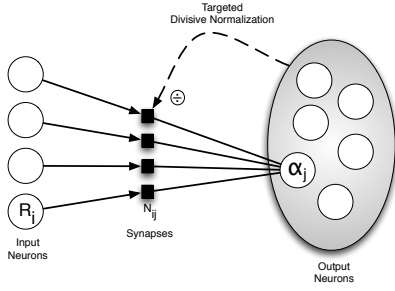
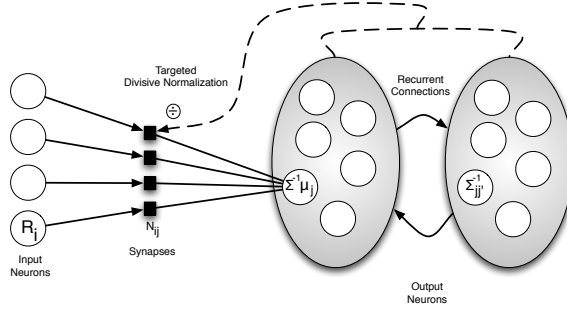

Figure 4: The LDA network model. Dendritically targeted inhibition is pooled from the activity of all neurons in the output layer and acts divisively.

Figure 5: DDM network model also includes recurrent connections which target the soma with both a linear excitatory signal and an inhibitory signal that also takes the form of a divisive normalization.

intensities of latent causes and that there are no correlations between the intensities of latent causes in temporally adjacent trials or scenes. This makes LDA a rather poor computational model for a task like olfactory foraging which requires the animal to track the rise a fall of odor intensities as it navigates its environment. We can model this more complicated task by replacing the static cause or odor intensity parameters with dynamic odor intensity parameters whose behavior is governed by an exponentiated Ornstein-Uhlenbeck process with drift and diffusion matrices given by ($\Lambda$ and $\Sigma_D$). We call this variant of LDA the Dynamic Document Model (DDM) as it could be used to model smooth changes in the distribution of topics over the course of a single document.

## 5.1 DDM Model

Thus the generative model for the DDM is as follows:

1. For latent cause $k = 1, \ldots, K$,
   (a) Cause distribution over spikes $\beta_k \sim \text{Dirichlet}(\eta_0)$
2. For scene $t = 1, \ldots, T$,
   (a) Log intensity of causes $\mathbf{c}(t) \sim \text{Normal}(\mathbf{\Lambda c}_{t-1}, \mathbf{\Sigma}_D)$
   (b) Number of spikes in neuron $j$ resulting from cause $k$,
       $N_{j,k}(t) \sim \text{Poisson}(\beta_{j,k} \exp c_k(t))$
   (c) Number of spikes in neuron $j$, $R_j(t) = \sum_k N_{j,k}(t)$

This model bears many similarities to the Correlated and Dynamic topic models[22], but models dynamics over a short time scale, where the dynamic relationship $(\mathbf{\Lambda}, \mathbf{\Sigma}_D)$ is important.

## 5.2 Network Implementation

Once again the quantity of interest is the current distribution of latent causes, $p(\mathbf{c}(t)|\mathbf{R}(\tau), \tau = 0..T)$. If no spikes occur then no evidence is presented and posterior inference over $\mathbf{c}(t)$ is simply given by an undriven Kalman filter with parameters $(\mathbf{\Lambda}, \mathbf{\Sigma}_D)$. A recurrent neural network which uses a linear PPC to encode a posterior that evolves according to a Kalman filter has the property that neural responses are linearly related to the inverse covariance matrix of the posterior as well as that inverse covariance matrix times the posterior mean. In the absence of evidence, it is easy to show that these quantities must evolve according to recurrent dynamics which implement divisive normalization[10]. Thus, the patterns of neural activity which linearly encode them must do so as well. When a new spike arrives, optimal inference is no longer possible and a variational approximation must be utilized. As is shown in the supplement, this variational approximation is similar to the variational approximation used for LDA. As a result, a network which can divisively inhibit its synapses is able to implement approximate Bayesian inference. Curiously, this implies that the addition of spatial and temporal correlations to the latent causes adds very little complexity to the VB-PPC network implementation of probabilistic inference. All that is required is an additional inhibitory population which targets the somata in the output population. See Fig. 5.

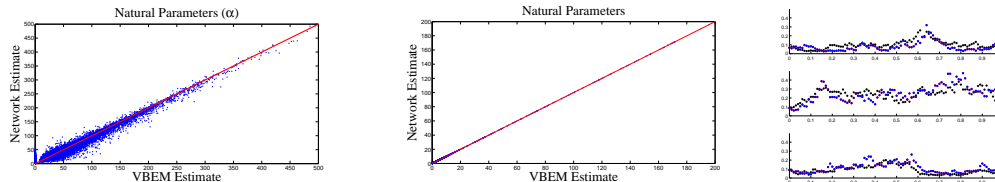

Figure 6: (Left) Neural network approximation to the natural parameters of the posterior distribution over topics (the $\alpha$'s) as a function of the VBEM estimate of those same parameters for a variety of 'documents'. (Center) Same as left, but for the natural parameters of the DDM (i.e the entries of the matrix $\mathbf{\Sigma}^{-1}(t)$ and $\mathbf{\Sigma}^{-1}\boldsymbol{\mu}(t)$ of the distribution over log topic intensities. (Right) Three example traces for cause intensity in the DDM. Black shows true concentration, blue and red (indistinguishable) show MAP estimates for the network and VBEM algorithms.

## 6 Experimental Results

We compared the PPC neural network implementations of the variational inference with the standard VBEM algorithm. This comparison is necessary because the two algorithms are not guaranteed to converge to the same solution due to the fact that we only required that the neural network dynamics have the same fixed points as the standard VBEM algorithm. As a result, it is possible for the two algorithms to converge to different local minima of the KL divergence. For the network implementation of LDA we find good agreement between the neural network and VBEM estimates of the natural parameters of the posterior. See Fig. 6(left) which shows the two algorithms estimates of the shape parameter of the posterior distribution over topic (odor) concentrations (a quantity which is proportional to the expected concentration). This agreement, however, is not perfect, especially when posterior predicted concentrations are low. In part, this is due to the fact we are presenting the network with difficult inference problems for which the true posterior distribution over topics (odors) is highly correlated and multimodal. As a result, the objective function (KL divergence) is littered with local minima. Additionally, the discrete iterations of the VBEM algorithm can take very large steps in the space of natural parameters while the neural network implementation cannot. In contrast, the network implementation of the DDM is in much better agreement with the VBEM estimation. See Fig. 6(right). This is because the smooth temporal dynamics of the topics eliminate the need for the VBEM algorithm to take large steps. As a result, the smooth network dynamics are better able to accurately track the VBEM algorithms output. For simulation details please see the supplement.

## 7 Discussion and Conclusion

In this work we presented a general framework for inference and learning with linear Probabilistic Population codes. This framework takes advantage of the fact that the Variational Bayesian Expectation Maximization algorithm generates approximate posterior distributions which are in an exponential family form. This is precisely the form needed in order to make probability distributions representable by a linear PPC. We then outlined a general means by which one can build a neural network implementation of the VB algorithm using this kind of neural code. We applied this VB-PPC framework to generate a biologically plausible neural network for spike train demixing. We chose this problem because it has many of the features of the canonical problem faced by nearly every layer of cortex, i.e. that of inferring the latent causes of complex mixtures of spike trains in the layer below. Curiously, this very complicated problem of probabilistic inference and learning ended up having a remarkably simple network solution, requiring only that neurons be capable of implementing divisive normalization via dendritically targeted inhibition and superlinear facilitation. Moreover, we showed that extending this approach to the more complex dynamic case in which latent causes change in intensity over time does not substantially increase the complexity of the neural circuit. Finally, we would like to note that, while we utilized a rate coding scheme for our linear PPC, the basic equations would still apply to any spike based log probability codes such as that considered Beorlin and Deneve[23].

# References

[1] Daniel Kersten, Pascal Mamassian, and Alan Yuille. Object perception as Bayesian inference. *Annual review of psychology*, 55:271–304, January 2004.

[2] Marc O Ernst and Martin S Banks. Humans integrate visual and haptic information in a statistically optimal fashion. *Nature*, 415(6870):429–33, 2002.

[3] Yair Weiss, Eero P Simoncelli, and Edward H Adelson. Motion illusions as optimal percepts. *Nature neuroscience*, 5(6):598–604, 2002.

[4] P N Sabes. The planning and control of reaching movements. *Current opinion in neurobiology*, 10(6): 740–6, 2000.

[5] Konrad P Körding and Daniel M Wolpert. Bayesian integration in sensorimotor learning. *Nature*, 427 (6971):244–7, 2004.

[6] Emanuel Todorov. Optimality principles in sensorimotor control. *Nature neuroscience*, 7(9):907–15, 2004.

[7] Erno Téglás, Edward Vul, Vittorio Girotto, Michel Gonzalez, Joshua B Tenenbaum, and Luca L Bonatti. Pure reasoning in 12-month-old infants as probabilistic inference. *Science (New York, N.Y.)*, 332(6033): 1054–9, 2011.

[8] W.J. Ma, J.M. Beck, P.E. Latham, and A. Pouget. Bayesian inference with probabilistic population codes. *Nature Neuroscience*, 2006.

[9] Jeffrey M Beck, Wei Ji Ma, Roozbeh Kiani, Tim Hanks, Anne K Churchland, Jamie Roitman, Michael N Shadlen, Peter E Latham, and Alexandre Pouget. Probabilistic population codes for Bayesian decision making. *Neuron*, 60(6):1142–52, 2008.

[10] J. M. Beck, P. E. Latham, and a. Pouget. Marginalization in Neural Circuits with Divisive Normalization. *Journal of Neuroscience*, 31(43):15310–15319, 2011.

[11] Tianming Yang and Michael N Shadlen. Probabilistic reasoning by neurons. *Nature*, 447(7148):1075–80, 2007.

[12] RHS Carpenter and MLL Williams. Neural computation of log likelihood in control of saccadic eye movements. *Nature*, 1995.

[13] Arnulf B a Graf, Adam Kohn, Mehrdad Jazayeri, and J Anthony Movshon. Decoding the activity of neuronal populations in macaque primary visual cortex. *Nature neuroscience*, 14(2):239–45, 2011.

[14] HB Barlow. Pattern Recognition and the Responses of Sensory Neurons. *Annals of the New York Academy of Sciences*, 1969.

[15] Wei Ji Ma, Vidhya Navalpakkam, Jeffrey M Beck, Ronald Van Den Berg, and Alexandre Pouget. Behavior and neural basis of near-optimal visual search. *Nature Neuroscience*, (May), 2011.

[16] DJ Heeger. Normalization of cell responses in cat striate cortex. *Visual Neuroscience*, 9, 1992.

[17] M Carandini, D J Heeger, and J a Movshon. Linearity and normalization in simple cells of the macaque primary visual cortex. *The Journal of neuroscience : the official journal of the Society for Neuroscience*, 17(21):8621–44, 1997.

[18] D. Blei, A. Ng, and M. Jordan. Latent Dirichlet Allocation. *JMLR*, 2003.

[19] M. Beal. *Variational Algorithms for Approximate Bayesian Inference*. PhD thesis, Gatsby Unit, UCL, 2003.

[20] D D Lee and H S Seung. Learning the parts of objects by non-negative matrix factorization. *Nature*, 401 (6755):788–91, 1999.

[21] M. Hoffman, D. Blei, and F. Bach. Online learning for Latent Dirichlet Allocation. In *NIPS*, 2010.

[22] D. Blei and J. Lafferty. Dynamic topic models. In *ICML*, 2006.

[23] M. Boerlin and S. Deneve. Spike-based population coding and working memory. *PLOS computational biology*, 2011.

